# Fast Training of Support Vector Classifiers

**F. Pérez-Cruz†, P. L. Alarcón-Diana†, A. Navia-Vázquez‡and A. Artés-Rodríguez‡.**
†Dpto. Teoría de la Señal y Com., Escuela Politécnica, Universidad de Alcalá.
28871-Alcalá de Henares (Madrid) Spain. e-mail: fernando@tsc.uc3m.es
‡Dpto. Tecnologías de las comunicaciones, Escuela Politécnica Superior,
Universidad Carlos III de Madrid, Avda. Universidad 30, 28911-Leganes (Madrid) Spain.

## Abstract

In this communication we present a new algorithm for solving Support Vector Classifiers (SVC) with large training data sets. The new algorithm is based on an Iterative Re-Weighted Least Squares procedure which is used to optimize the SVC. Moreover, a novel sample selection strategy for the working set is presented, which randomly chooses the working set among the training samples that do not fulfill the stopping criteria. The validity of both proposals, the optimization procedure and sample selection strategy, is shown by means of computer experiments using well-known data sets.

## 1 INTRODUCTION

The Support Vector Classifier (SVC) is a powerful tool to solve pattern recognition problems [13, 14] in such a way that the solution is completely described as a linear combination of several training samples, named the Support Vectors. The training procedure for solving the SVC is usually based on Quadratic Programming (QP) which presents some inherent limitations, mainly the computational complexity and memory requirements for large training data sets. This problem is typically avoided by dividing the QP problem into sets of smaller ones [6, 1, 7, 11], that are iteratively solved in order to reach the SVC solution for the whole set of training samples. These schemes rely on an optimizing engine, QP, and in the sample selection strategy for each sub-problem, in order to obtain a fast solution for the SVC.

An Iterative Re-Weighted Least Squares (IRWLS) procedure has already been proposed as an alternative solver for the SVC [10] and the Support Vector Regressor [9], being computationally efficient in absolute terms. In this communication, we will show that the IRWLS algorithm can replace the QP one in any chunking scheme in order to find the SVC solution for large training data sets. Moreover, we consider that the strategy to decide which training samples must join the working set is critical to reduce the total number of iterations needed to attain the SVC solution, and the runtime complexity as a consequence. To aim for this issue, the computer program $SVC^{radit}$ have been developed so as to solve the SVC for large training data sets using IRWLS procedure and fixed-size working sets.

The paper is organized as follows. In Section 2, we start by giving a summary of the IRWLS procedure for SVC and explain how it can be incorporated to a chunking scheme to obtain an overall implementation which efficiently deals with large training data sets.

We present in Section 3 a novel strategy to make up the working set. Section 4 shows the capabilities of the new implementation and they are compared with the fastest available SVC implementation, $SVM^{light}$ [6]. We end with some concluding remarks.

## 2   IRWLS-SVC

In order to solve classification problems, the SVC has to minimize

$$L_P \equiv \frac{1}{2}\|\mathbf{w}\|^2 + C\sum_i \xi_i - \sum_i \mu_i \xi_i - \sum_i \alpha_i(y_i(\phi(\mathbf{x}_i)^T\mathbf{w}+b)-1+\xi_i) \quad (1)$$

with respect to $\mathbf{w}$, $b$ and $\xi_i$ and maximize it with respect to $\alpha_i$ and $\mu_i$, subject to $\alpha_i, \mu_i \geq 0$, where $\phi(\cdot)$ is a nonlinear transformation (usually unknown) to a higher dimensional space and C is a penalization factor. The solution to (1) is defined by the Karush-Kuhn-Tucker (KKT) conditions [2]. For further details on the SVC, one can refer to the tutorial survey by Burges [2] and to the work of Vapnik [13, 14].

In order to obtain an IRWLS procedure we will first need to rearrange (1) in such a way that the terms depending on $\xi_i$ can be removed because, at the solution $C - \alpha_i - \mu_i = 0$ $\quad \forall i$ (one of the KKT conditions [2]) must hold.

$$
\begin{aligned}
L_P &= \frac{1}{2}\|\mathbf{w}\|^2 + \sum_i \alpha_i(1 - y_i(\phi^T(\mathbf{x}_i)\mathbf{w}+b)) \\
&= \frac{1}{2}\|\mathbf{w}\|^2 + \frac{1}{2}\sum_i \frac{2\alpha_i}{1 - y_i(\phi^T(\mathbf{x}_i)\mathbf{w}+b)}\left(y_i - (\phi^T(\mathbf{x}_i)\mathbf{w}+b)\right)^2 \\
&= \frac{1}{2}\|\mathbf{w}\|^2 + \frac{1}{2}\sum_i a_i e_i^2 \quad (2)
\end{aligned}
$$

where

$$e_i = y_i - (\phi^T(\mathbf{x}_i)\mathbf{w}+b) \quad \text{and} \quad a_i = \frac{2\alpha_i}{1 - y_i(\phi^T(\mathbf{x}_i)\mathbf{w}+b)}$$

The weighted least square nature of (2) can be understood if $e_i$ is defined as the error on each sample and $a_i$ as its associated weight, where $\frac{1}{2}\|\mathbf{w}\|^2$ is a regularizing functional. The minimization of (2) cannot be accomplished in a single step because $a_i = a_i(e_i)$, and we need to apply an IRWLS procedure [4], summarized below in tree steps:

1. Considering the $a_i$ fixed, minimize (2).

2. Recalculate $a_i$ from the solution on step 1.

3. Repeat until convergence.

In order to work with Reproducing Kernels in Hilbert Space (RKHS), as the QP procedure does, we require that $\mathbf{w} = \sum_i \beta_i y_i \phi(\mathbf{x}_i)$ and in order to obtain a non-zero $b$, that $\sum_i \beta_i y_i = 0$. Substituting them into (2), its minimum with respect to $\beta_i$ and $b$ for a fixed set of $a_i$ is found by solving the following linear equation system[1]

$$\begin{bmatrix} \mathbf{H} + \mathbf{D_a}^{-1} & \mathbf{y} \\ \mathbf{y}^T & 0 \end{bmatrix}\begin{bmatrix} \beta \\ b \end{bmatrix} = \begin{bmatrix} \mathbf{1} \\ 0 \end{bmatrix} \quad (3)$$

where

$$\mathbf{y} = [y_1, y_2, \ldots y_n]^T \tag{4}$$

$$(\mathbf{H})_{ij} = y_i y_j \phi^T(\mathbf{x}_i)\phi(\mathbf{x}_j) = y_i y_j K(\mathbf{x}_i, \mathbf{x}_j) \qquad \forall i, j = 1, \ldots, n \tag{5}$$

$$(\mathbf{D_a})_{ij} = a_i \delta[i - j] \qquad \forall i, j = 1, \ldots, n \tag{6}$$

$$\boldsymbol{\beta} = [\beta_1, \beta_2, \ldots, \beta_n]^T \tag{7}$$

and $\delta[\cdot]$ is the discrete impulse function. Finally, the dependency of $a_i$ upon the Lagrange multipliers is eliminated using the KKT conditions, obtaining

$$a_i = \begin{cases} 0, & e_i y_i < 0 \\ \frac{2C}{e_i y_i}, & y_i e_i \geq 0 \end{cases} \tag{8}$$

## 2.1 IRWLS ALGORITHMIC IMPLEMENTATION

The SVC solution with the IRWLS procedure can be simplified by dividing the training samples into three sets. The first set, $S_1$, contains the training samples verifying $0 < \beta_i < C$, which have to be determined by solving (3). The second one, $S_2$, includes every training sample whose $\beta_i = 0$. And the last one, $S_3$, is made up of the training samples whose $\beta_i = C$. This division in sets is fully justified in [10]. The IRWLS-SVC algorithm is shown in Table 1.

---

0. Initialization:

$S_1$ will contain every training sample, $S_2 = \emptyset$ and $S_3 = \emptyset$. Compute $\mathbf{H}$. $\mathbf{e\_a} = \mathbf{y}$, $\boldsymbol{\beta\_a} = 0$, $b\_a = 0$, $\mathbf{G}_{13} = \mathbf{G}_{in}$, $\mathbf{a} = \mathbf{1}$ and $G_{b3} = G_{b_{in}}$.

1. Solve $\begin{bmatrix} (\mathbf{H})_{S_1,S_1} + \mathbf{D}_{(\mathbf{a})S_1}^{-1} & (\mathbf{y})_{S_1} \\ (\mathbf{y})_{S_1}^T & 0 \end{bmatrix} \begin{bmatrix} (\boldsymbol{\beta})_{S_1} \\ b \end{bmatrix} = \begin{bmatrix} \mathbf{1} - \mathbf{G}_{13} \\ G_{b3} \end{bmatrix}$,

$(\boldsymbol{\beta})_{S_2} = 0$ and $(\boldsymbol{\beta})_{S_3} = C$

2. $\mathbf{e} = \mathbf{e\_a} - \mathbf{D_y}\mathbf{H}(\boldsymbol{\beta} - \boldsymbol{\beta\_a}) - (b - b\_a)\mathbf{1}$

3. $a_i = \begin{cases} 0, & e_i y_i < 0 \\ \frac{2C}{e_i y_i}, & e_i y_i \geq 0 \end{cases} \forall i \in S_1 \cup S_2 \cup S_3$

4. Sets reordering:

   a. Move every sample in $S_3$ with $e_i y_i < 0$ to $S_2$.

   b. Move every sample in $S_1$ with $\beta_i = C$ to $S_3$.

   c. Move every sample in $S_1$ with $a_i = 0$ to $S_2$.

   d. Move every sample in $S_2$ with $a_i \neq 0$ to $S_1$.

5. $\mathbf{e\_a} = \mathbf{e}$, $\boldsymbol{\beta\_a} = \boldsymbol{\beta}$, $\mathbf{G}_{13} = (\mathbf{H})_{S_1,S_3}(\boldsymbol{\beta})_{S_3} + (\mathbf{G}_{in})_{S_1}$,

   $b\_a = b$ and $G_{b3} = -\mathbf{y}_{S_3}^T(\boldsymbol{\beta})_{S_3} + G_{b_{in}}$.

6. Go to step 1 and repeat until convergence.

---

Table 1: IRWLS-SVC algorithm.

The IRWLS-SVC procedure has to be slightly modified in order to be used inside a chunking scheme as the one proposed in [8, 6], such that it can be directly applied in the one proposed in [1]. A chunking scheme is needed to solve the SVC whenever $\mathbf{H}$ is too large to fit into memory. In those cases, several SVC with a reduced set of training samples are iteratively solved until the solution for the whole set is found. The samples are divide into a working set, $S_w$, which is solved as a full SVC problem, and an inactive set, $S_{in}$. If there are support vectors in the inactive set, as it might be, the inactive set modifies the IRWLS-SVC procedure, adding a contribution to the independent term in the linear equation system (3). Those support vectors in $S_{in}$ can be seen as anchored samples in $S_3$, because their $\beta_i$ is

not zero and can not be modified by the IRWLS procedure. Then, such contribution ($\mathbf{G}_{in}$ and $G_{b_{in}}$) will be calculated as $\mathbf{G}_{13}$ and $G_{b3}$ are (Table 1, $5^{th}$ step), before calling the IRWLS-SVC algorithm. We have already modified the IRWLS-SVC in Table 1 to consider $\mathbf{G}_{in}$ and $G_{b_{in}}$, which must be set to zero if the Hessian matrix, $\mathbf{H}$, fits into memory for the whole set of training samples.

The resolution of the SVC for large training data sets, employing as minimization engine the IRWLS procedure, is summarized in the following steps:

1. Select the samples that will form the working set.
2. Construct $\mathbf{G}_{in} = (\mathbf{H})_{S_w, S_{in}} (\beta)_{S_{in}}$ and $G_{b_{in}} = -\mathbf{y}_{S_{in}}^T (\beta)_{S_{in}}$
3. Solve the IRWLS-SVC procedure, following the steps in Table 1.
4. Compute the error of every training sample.
5. If the stopping conditions

$$y_i e_i < \varepsilon \qquad \forall i | \quad \beta_i = 0 \qquad \qquad (9)$$
$$e_i y_i > -\varepsilon \qquad \forall i | \quad \beta_i = C \qquad \qquad (10)$$
$$|e_i y_i| < \varepsilon \qquad \forall i | \quad 0 < \beta_i < C \qquad (11)$$

are fulfilled, the SVC solution has been reached.

The stopping conditions are the ones proposed in [6] and $\varepsilon$ must be a small value around $10^{-3}$, a full discussion concerning this topic can be found in [6].

## 3 SAMPLE SELECTION STRATEGY

The selection of the training samples that will constitute the working set in each iteration is the most critical decision in any chunking scheme, because such decision is directly involved in the number of IRWLS-SVC (or QP-SVC) procedures to be called and in the number of reproducing kernel evaluations to be made, which are, by far, the two most time consuming operations in any chunking schemes.

In order to solve the SVC efficiently, we first need to define a candidate set of training samples to form the working set in each iteration. The candidate set will be made up, as it could not be otherwise, with all the training samples that violate the stopping conditions (9)-(11); and we will also add all those training samples that satisfy condition (11) but a small variation on their error will make them violate such condition.

The strategies to select the working set are as numerous as the number of problems to be solved, but one can think three different simple strategies:

- Select those samples which do not fulfill the stopping criteria and present the largest $|e_i|$ values.
- Select those samples which do not fulfill the stopping criteria and present the smallest $|e_i|$ values.
- Select them randomly from the ones that do not fulfill the stopping conditions.

The first strategy seems the more natural one and it was proposed in [6]. If the largest $|e_i|$ samples are selected we guanrantee that attained solution gives the greatest step towards the solution of (1). But if the step is too large, which usually happens, it will cause the solution in each iteration and the $\beta_i$ values to oscillate around its optimal value. The magnitude of this effect is directly proportional to the value of $C$ and $q$ (size of the working set), so in the case of small $C$ ($C < 10$) and low $q$ ($q < 20$) it would be less noticeable.

The second one is the most conservative strategy because we will be moving towards the solution of (1) with small steps. Its drawback is readily discerned if the starting point is inappropriate, needing too many iterations to reach the SVC solution.

The last strategy, which has been implemented together with the IRWLS-SVC procedure, is a mid-point between the other two, but if the number of samples whose $0 < \beta_i < C$ increases above $q$ there might be some iterations where we will make no progress (working set is only made up of the training samples that fulfill the stopping condition in (11)). This situation is easily avoided by introducing one sample that violates each one of the stopping conditions per class. Finally, if the cardinality of the candidate set is less than $q$ the working set is completed with those samples that fulfil the stopping criteria conditions and present the least $|e_i|$.

In summary, the sample selection strategy proposed is[2]:

1. Construct the candidate set, $S_c$ with those samples that do not fulfill stopping conditions (9) and (10), and those samples whose $\beta$ obeys $0 < \beta_i < C$.

2. If $|S_c| < n$ go to 5.

3. Choose a sample per class that violates each one of the stopping conditions and move them from $S_c$ to the working set, $S_w$.

4. Choose randomly $n - |S_w|$ samples from $S_c$ and move then to $S_w$. Go to Step 6.

5. Move every sample form $S_c$ to $S_w$ and the $n - |S_w|$ samples that fulfill the stopping conditions (9) and (10) and present the lowest $|e_i|$ values are used to complete $S_w$.

6. Go on, obtaining $\mathbf{G}_{in}$ and $G_{b_{in}}$.

## 4   BENCHMARK FOR THE IRWLS-SVC

We have prepared two different experiments to test both the IRWLS and the sample selection strategy for solving the SVC. The first one compares the IRWLS against QP and the second one compares the samples selection strategy, together with the IRWLS, against a complete solving procedure for SVC, the $SVM^{light}$.

In the first trial, we have replaced the LOQO interior point optimizer used by $SVM^{light}$ version 3.02 [5] by the IRWLS-SVC procedure in Table 1, to compare both optimizing engines with equal samples selection strategy. The comparison has been made over a Pentium III-450MHz with 128Mb running on Window98 and the programs have been compiled using Microsoft Developer 6.0. In Table 2, we show the results for two data sets: the first

| | Adult4 4781 | | | | Splice 2175 | | | |
|---|---|---|---|---|---|---|---|---|
| | CPU time | | Optimize Time | | CPU time | | Optimize Time | |
| $q$ | LOQO | IRWLS | LOQO | IRWLS | LOQO | IRWLS | LOQO | IRWLS |
| 20 | 21.25 | 20.70 | 0.61 | 0.39 | 46.19 | 30.76 | 21.94 | 4.77 |
| 40 | 20.60 | 19.22 | 1.01 | 0.17 | 71.34 | 24.93 | 46.26 | 8.07 |
| 70 | 21.15 | 18.72 | 2.30 | 0.46 | 53.77 | 20.32 | 34.24 | 7.72 |

Table 2: CPU Time indicates the consume time in seconds for the whole procedure. The Optimize Time indicates the consume time in second for the LOQO or IRWLS procedure.

one, containing 4781 training samples, needs most CPU resources to compute the RKHS and the second one, containing 2175 training samples, uses most CPU resources to solve the SVC for each $S_w$, where $q$ indicates the size of the working set. The value of C has

been set to 1 and 1000, respectively, and a Radial Basis Function (RBF) RKHS [2] has been employed, where its parameter $\sigma$ has been set, respectively, to 10 and 70.

As it can be seen, the $SVM^{light}$ with IRWLS is significantly faster than the LOQO procedure in all cases. The kernel cache size has been set to 64Mb for both data sets and for both procedures. The results in Table 2 validates the IRWLS procedure as the fastest SVC solver.

For the second trial, we have compiled a computer program that uses the IRWLS-SVC procedure and the working set selection in Section 3, we will refer to it as $SVC^{radit}$ from now on. We have borrowed the chunking and shrinking ideas from the $SVM^{light}$ [6] for our computer program. To test these two programs several data sets have been used. The Adult and Web data sets have been obtained from J. Platt's web page http://research.microsoft.com/ˉjplatt/smo.html/; the Gauss-M data set is a two dimensional classification problem proposed in [3] to test neural networks, which comprises a gaussian random variable for each class, which highly overlap. The Banana, Diabetes and Splice data sets have been obtained from Gunnar Rätsch web page http://svm.first.gmd.de/ˉraetsch/. The selection of $C$ and the RKHS has been done as indicated in [11] for Adult and Web data sets and in http://svm.first.gmd.de/ˉraetsch/ for Banana, Diabetes and Splice data sets. In Table 3, we show the runtime complexity for each data set, where the value of $q$ has been elected as the one that reduces the runtime complexity.

| Database | Dim | N Sampl. | C | $\sigma$ | SV | q radit | q light | CPU time radit | CPU time light |
|---|---|---|---|---|---|---|---|---|---|
| Adult6 | 123 | 11221 | 1 | 10 | 4477 | 150 | 40 | 118.2 | 124.46 |
| Adult9 | 123 | 32562 | 1 | 10 | 12181 | 130 | 70 | 1093.29 | 1097.09 |
| Adult1 | 123 | 1605 | 1000 | 10 | 630 | 100 | 10 | 25.98 | 113.54 |
| Web1 | 300 | 2477 | 5 | 10 | 224 | 100 | 10 | 2.42 | 2.36 |
| Web7 | 300 | 24693 | 5 | 10 | 1444 | 150 | 10 | 158.13 | 124.57 |
| Gauss-M | 2 | 4000 | 1 | 1 | 1736 | 70 | 10 | 12.69 | 48.28 |
| Gauss-M | 2 | 4000 | 100 | 1 | 1516 | 100 | 10 | 61.68 | 3053.20 |
| Banana | 2 | 400 | 316.2 | 1 | 80 | 40 | 70 | 0.33 | 0.77 |
| Banana | 2 | 4900 | 316.2 | 1 | 1084 | 70 | 40 | 22.46 | 1786.56 |
| Diabetes | 8 | 768 | 10 | 2 | 409 | 40 | 10 | 2.41 | 6.04 |
| Splice | 69 | 2175 | 1000 | 70 | 525 | 150 | 20 | 14.06 | 49.19 |

Table 3: Several data sets runtime complexity, when solved with the $SVC^{radit}$, $radit$ for short, and $SVM^{light}$, $light$ for short.

One can appreciate that the $SVC^{radit}$ is faster than the $SVM^{light}$ for most data sets. For the Web data set, which is the only data set the $SVM^{light}$ is sligthly faster, the value of $C$ is low and most training samples end up as support vector with $\beta_i < C$. In such cases the best strategy is to take the largest step towards the solution in every iteration, as the $SVM^{light}$ does [6], because most training samples $\beta_i$ will not be affected by the others training samples $\beta_j$ value. But in those case the value of $C$ increases the $SVC^{radit}$ samples selection strategy is a much more appropriate strategy than the one used in $SVM^{light}$.

## 5 CONCLUSIONS

In this communication a new algorithm for solving the SVC for large training data sets has been presented. Its two major contributions deal with the optimizing engine and the sample selection strategy. An IRWLS procedure is used to solve the SVC in each step, which is much faster that the usual QP procedure, and simpler to implement, because the

most difficult step is the linear equation system solution that can be easily obtained by LU decomposition means [12]. The random working set selection from the samples not fulfilling the KKT conditions is the best option if the working is be large, because it reduces the number of chunks to be solved. This strategy benefits from the IRWLS procedure, which allows to work with large training data set. All these modifications have been concreted in the $SVC^{radit}$ solving procedure, publicly available at http://svm.tsc.uc3m.es/.

# 6 ACKNOWLEDGEMENTS

We are sincerely grateful to Thorsten Joachims who has allowed and encouraged us to use his $SVM^{light}$ to test our IRWLS procedure, comparisons which could not have been properly done otherwise.

## Footnotes

[1]The detailed description of the steps needed to obtain (3) from (2) can be found in [10].

[2]In what follows, $|\cdot|$ represents absolute value for numbers and cardinality for sets

# References

[1] B. E. Boser, I. M. Guyon, and V. Vapnik. A training algorithm for optimal margin classifiers. In *5th Annual Workshop on Computational Learning Theory*, Pittsburg, U.S.A., 1992.

[2] C. J. C. Burges. A tutorial on support vector machines for pattern recognition. *Data Mining and Knowledge Discovery*, 2(2):121–167, 1998.

[3] S. Haykin. *Neural Networks: A comprehensive foundation*. Prentice-Hall, 1994.

[4] P. W. Holland and R. E. Welch. Robust regression using iterative re-weighted least squares. *Communications of Statistics Theory Methods*, A6(9):813–27, 1977.

[5] T. Joachims. http://www-ai.informatik.uni-dortmund.de /forschung/verfahren /svm_light /svm_light.eng.html. Technical report, University of Dortmund, Informatik, AI-Unit Collaborative Research Center on 'Complexity Reduction in Multivariate Data', 1998.

[6] T. Joachims. Making Large Scale SVM Learning Practical, In *Advances in Kernel Methods— Support Vector Learning*, Editors Schölkopf, B., Burges, C. J. C. and Smola, A. J., pages 169–184. M.I.T. Press, 1999.

[7] E. Osuna, R. Freund, and F. Girosi. An improved training algorithm for support vector machines. In *Proc. of the 1997 IEEE Workshop on Neural Networks for Signal Processing*, pages 276–285, Amelia Island, U.S.A, 1997.

[8] E. Osuna and F. Girosi. Reducing the run–time complexity of support vector machines. In *ICPR'98*, Brisbane, Australia, August 1998.

[9] F. Pérez-Cruz, A. Navia-Vázquez, , P. L. Alarcón-Diana, and A. Artés-Rodríguez. An irwls proceure for svr. In *the Proceedings of the EUSIPCO'00*, Tampere, Finland, 9 2000.

[10] F. Pérez-Cruz, A. Navia-Vázquez, J. L. Rojo-Álvarez, and A. Artés-Rodríguez. A new training algorithm for support vector machines. In *Proceedings of the Fifth Bayona Workshop on Emerging Technologies in Telecommunications*, volume 1, pages 116–120, Baiona, Spain, 9 1999.

[11] J. C. Platt. Sequential Minimal Optimization: A Fast Algorithm for Training Suppor Vector Machines, In *Advances in Kernel Methods— Support Vector Learning*, Editors Schölkopf, B., Burges, C. J. C. and Smola, A. J., pages 185–208. M.I.T. Press, 1999.

[12] W. H. Press, S. A. Teukolsky, W. T. Vetterling, and B. P. Flannery. *Numerical Recipes in C*. Cambridge University Press, Cambridge, UK, 2 edition, 1994.

[13] V. N. Vapnik. *The Nature of Statistical Learning Theory*. Springer–Verlag, 1995.

[14] V. N. Vapnik. *Statistical Learning Theory*. John Wiley & Sons, 1998.
